# Probabilistic Semantic Video Indexing

**Milind R. Naphade, Igor Kozintsev and Thomas Huang**
Department of Electrical and Computer Engineering
University of Illinois at Urbana-Champaign
{*milind,igor,huang*}*@ifp.uiuc.edu*

## Abstract

We propose a novel probabilistic framework for semantic video indexing. We define probabilistic multimedia objects (multijects) to map low-level media features to high-level semantic labels. A graphical network of such multijects (multinet) captures scene context by discovering intra-frame as well as inter-frame dependency relations between the concepts. The main contribution is a novel application of a factor graph framework to model this network. We model relations between semantic concepts in terms of their co-occurrence as well as the temporal dependencies between these concepts within video shots. Using the sum-product algorithm [1] for approximate or exact inference in these factor graph multinets, we attempt to correct errors made during isolated concept detection by forcing high-level constraints. This results in a significant improvement in the overall detection performance.

## 1 Introduction

Research in video retrieval has traditionally focussed on the paradigm of query-by-example (QBE) using low-level features [2]. Query by keywords/key-phrases (QBK) (preferably semantic) instead of examples has motivated recent research in semantic video indexing. For this, we need models which capture the feature representation corresponding to these keywords. A QBK system can support semantic retrieval for a small set of keywords and also act as the first step in QBE systems to narrow down the search. The difficulty lies in the gap between low-level media features and high-level semantics. Recent attempts to address this include detection of audio-visual events like explosion [3] and semantic visual templates [4].

We propose a statistical pattern recognition approach for training probabilistic multimedia objects (multijects) which map the high level concepts to low-level audio-visual features. We also propose a probabilistic factor graph framework, which models the interaction between concepts within each video frame as well as across the video frames within each video shot. Factor graphs provide an elegant framework to represent the stochastic relationship between concepts, while the sum-product algo-

rithm provides an efficient tool to perform learning and inference in factor graphs. Using exact as well as approximate inference (through loopy probability propagation) we show that there is significant improvement in the detection performance.

## 2    Proposed Framework

To support retrieval based on high-level queries like 'Explosion on a beach', we need models for the event explosion and site beach. User queries might similarly involve sky, helicopter, car-chase etc. Detection of some of these concepts may be possible, while some others may not be directly observable. To support such queries, we proposed a probabilistic multimedia object (multiject) [3] as shown in Figure 1 (a), which has a semantic label and which summarizes a time sequence of features from multiple media. A Multiject can belong to any of the three categories: objects (car, man, helicopter), sites (outdoor, beach), or events (explosion, man-walking).

Intuitively it is clear that the presence of certain multijects suggests a high possibility of detecting certain other multijects. Similarly some multijects are less likely to occur in the presence of others. The detection of sky and water boosts the chances of detecting a beach, and reduces the chances of detecting Indoor. It might also be possible to detect some concepts and infer more complex concepts based on their relation with the detected ones. Detection of human speech in the audio stream and a face in the video stream may lead to the inference of human talking. To integrate all the multijects and model their interaction, we propose the network of multijects which we term as multinet. A conceptual figure of a multinet is shown in Figure 1 (b) with positive (negative) signs indicating positive (negative) interaction. In

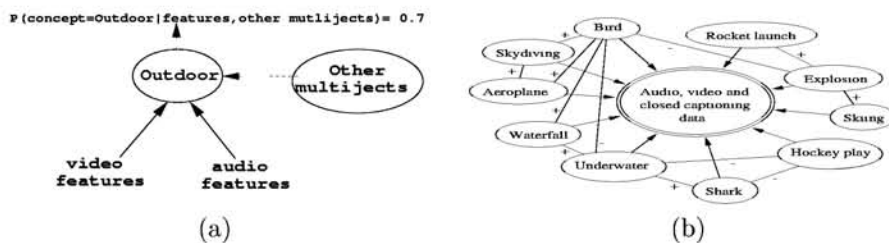

Figure 1: (a) A probabilistic multimedia object. (b) A conceptual multinet.

Section 5 we present a factor graph multinet implementation.

## 3    Video segmentation and Feature Extraction

We have digitized movies of different genres to create a large database of a few hours of video data. The video clips are segmented into shots using the algorithm in [5]. We then perform spatio-temporal segmentation [2] within each shot to obtain and track regions homogeneous in color and motion separated by strong edges. Large dominant regions are labeled manually. Each region is then processed to extract features characterizing the color (3-channel histogram [3]), texture (statistical properties of the Gray-level Co-occurrence matrices at 4 different orientations [6]), structure (edge direction histogram [7]), motion (affine motion parameters) and shape (moment invariants [8]). Details about the extracted features can be found in [9]. For sites we use color, texture and structural features (84 elements)

and for objects and events we use all features (98 elements)[1]. Audio features are extracted as in [10]. For training our multiject and multinet models we use 1800 frames from different video shots and for testing our framework we use 9400 frames. Since consecutive images within a shot are correlated, the video data is subsampled to create the training and testing without redundancy.

## 4    Modeling semantic concepts using Multijects

We use an identical approach to model concepts in video and audio (independently and jointly). The following site multijects are used in our experiments: *sky, water, forest, rocks* and *snow*. Audio-only multijects (*human-speech, music*) can be found in [10] and audio-visual multijects (*explosion*) in [3]. Detection of multijects is performed on every segmented region[2] within each video frame. Let the feature vector for the region $j$ be $\vec{X}_j$. We model the semantic concept as a binary random variable and define the two hypotheses $H_0$ and $H_1$ as

$$H_0 : \vec{X}_j \sim P_0(\vec{X}_j) \quad H_1 : \vec{X}_j \sim P_1(\vec{X}_j) \tag{1}$$

where $P_0(\vec{X}_j)$ and $P_1(\vec{X}_j)$ denote the class conditional probability density functions conditioned on the null hypothesis (concept absent) and the true hypothesis (concept present). $P_0(\vec{X}_j)$ and $P_1(\vec{X}_j)$ are modeled using a mixture of Gaussian components for the *site* multijects[3]. For *objects* and *events* (in video and audio), hidden Markov models replace the Gaussian mixture models and feature vectors for all the frames within a shot constitute to the time series modeled. The detection performance for the five *site* multijects on the test-set is given in Table 1.

| multiject | Rocks | Sky | Snow | Water | Forest |
|---|---|---|---|---|---|
| Detection (%) | 77 | 81.8 | 81.5 | 79.4 | 85.1 |
| False Alarm (%) | 24.1 | 11.9 | 12.9 | 15.6 | 14.9 |

Table 1: Maximum likelihood binary classification performance for *site* multijects.

### 4.1    Frame level semantic features

Since multijects are used as semantic feature detectors at a regional level, it is easy to define multiject-based semantic features at the frame level by integrating the region-level classification. We check each region for each concept individually and obtain probabilities of each concept being present or absent in the region. Imperfect segmentation does not hurt us too much since these soft decisions are modified in the multinet based on high-level constraints. Defining a binary random variable $R_{ij}$ ($R_{ij} = 1/0$ if concept present/absent) and assuming uniform priors on the presence or absence of a concept in any region we can use Bayes' rule to obtain:

$$P(R_{ij} = 1|\vec{X}_j) = P(\vec{X}_j|R_{ij} = 1)/(P(\vec{X}_j|R_{ij} = 1) + P(\vec{X}_j|R_{ij} = 0)) \tag{2}$$

Defining binary random variables $F_i$, $i \in \{1, N\}$ ($N$ is the number of concepts) to take on value 1 if concept $i$ is present in the frame and value 0 otherwise, we use the

$OR$ function to combine soft decisions for each concept from all regions to obtain $F_i$. Let $\mathcal{X} = \{\vec{X_1}, \ldots, \vec{X_M}\}$ ($M$ is the number of regions in a frame), then

$$P(F_i = 0|\mathcal{X}) = \prod_{j=1}^{j=M} P(R_{ij} = 0|\vec{X_j}) \ and \ P(F_i = 1|\mathcal{X}) = 1 - P(F_i = 0|\mathcal{X}) \quad (3)$$

## 5    The multinet as a factor graph

To model the interaction between multijects in a multinet, we propose to use a *factor graph* [1] framework. Factor graphs subsume graphical models like Bayesian nets and Markov random fields and have been successfully applied in the area of channel error correction coding [1] and specifically, iterative decoding. Let $\mathbf{x} = \{x_1, x_2, \ldots, x_n\}$ be a vector of variables. A *factor graph* visualizes the factorization of a global function $f(\mathbf{x})$. Let $f(\mathbf{x})$ factor as

$$f(\mathbf{x}) = \prod_{i=1} f_i(\mathbf{x}^{(i)}) \quad (4)$$

where $\mathbf{x}^{(i)}$ is the set of variables of the function $f_i$. A factor graph for $f$ is defined as the bipartite graph with two vertex classes $V_f$ and $V_v$ of sizes $m$ and $n$ respectively such that the $i$th node in $V_f$ is connected to the $j$th node in $V_v$ iff $f_i$ is a function of $x_j$. Figure 2 (a) shows a simple factor graph representation of $f(x, y, z) = f_1(x, y) f_2(y, z)$ with function nodes $f_1, f_2$ and variable nodes $x, y, z$.

Many signal processing and learning problems are formulated as optimizing a global function $f(\mathbf{x})$ marginalized for a subset of its arguments. The algorithm which allows us to perform this efficiently, though in most cases only approximately, is called the **sum-product algorithm**. The sum-product algorithm works by computing messages at the nodes using a simple rule and then passing the messages between nodes according to a reasonable schedule. A message from a function node to a variable node is the product of all messages incoming to the function node with the function itself, marginalized for the variable associated with the variable node. A message from a variable node to a function node is simply the product of all messages incoming to the variable node from other functions connected to it. Pearl's probability propagation working on a Bayesian net is equivalent to the sum-product algorithm applied to the corresponding factor graph. If the factor graph is a tree, exact inference is possible using a single set of forward and backward passage of messages. For all other cases inference is approximate and the message passing is iterative [1] leading to loopy probability propagation. This has a direct bearing on our problem because relations between semantic concepts are complicated and in general contain numerous cycles (e.g., see Figure 1 (b)).

### 5.1    Relating semantic concepts in a factor graph

We now describe a frame-level factor graph to model the probabilistic relations between various frame-level semantic features $F_i$ obtained using Equation 3. To capture the co-occurrence relationship between the five semantic concepts at the frame-level, we define a function node which is connected to the five variable nodes representing the concepts as shown in Figure 2 (b). This function node represents $P(F_1, F_2, F_3, .., F_N)$. The function nodes below the five variable nodes denote the messages passed by the OR function of Equation 3 ($P(F_i = 1)$, $P(F_i = 0)$). These are then propagated to the function node. At the function node the messages are

multiplied by the function which is estimated from the co-occurrence of the concepts in the training set. The function node then sends back messages summarized for each variable. This modifies the soft decisions at the variable nodes according to the high-level relationship between the five concepts. In general, the distribution

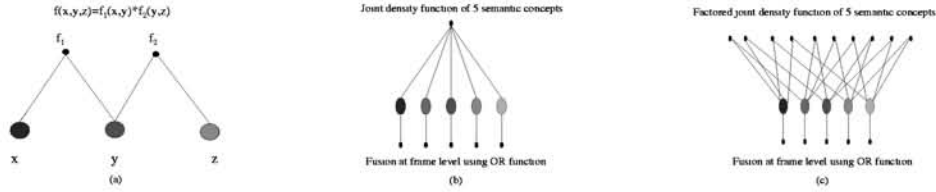

Figure 2: (a) An example of a simple factor graph (b)A multinet: Accounting for concept dependencies using a single function (b) Another multinet: Replacing the function in (b) by a product of 10 local functions.

at the function node in Figure 2 (b) is exponential in the number of concepts (N) and the computational cost may increase quickly. To alleviate this we can enforce a factorization of the function in Figure 2 (b) as a product of a set of local functions where each local function accounts for co-occurrence of two variables only. This modification to the graph in Figure 2 (b) is shown in Figure 2 (c). Each function in Figure 2 (c) represents the joint probability mass of those two variables that are its arguments (and there are $C_2^N$ such functions) thus reducing the complexity. The factor graph is no longer a tree and exact inference becomes hard as the number of loops grows. We then apply iterative techniques based on the sum-product algorithm to overcome this. We can also incorporate temporal

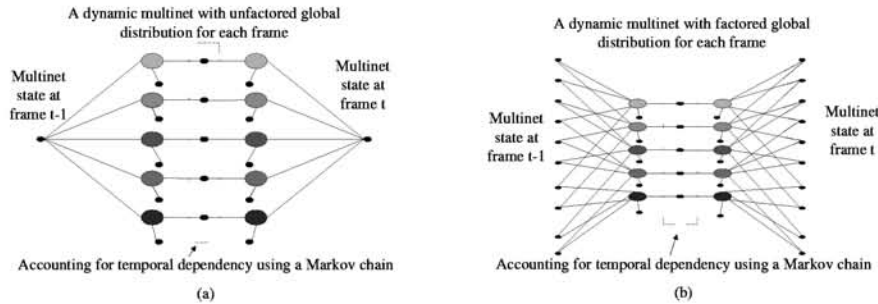

Figure 3: (a) Replicating the multinet in Figure 2 (b) for each frame in a shot and introducing temporal dependencies between the value of each concept in consecutive frames. (b) Repeating this for Figure 2 (c).

dependencies. This can be done by replicating the slice of factor graph in Figure 2 (b) or (c) as many times as the number of frames within a single video shot and by introducing a first order Markov chain for each concept. Figures 3 (a) and (b) show two consecutive time slices and extend the models in Figures 2 (b) and (c) respectively. The horizontal links in Figures 3 (a), (b) connect the variable node for each concept in a time slice to the corresponding variable node in the next time slice through a function modeling the transition probability. This framework now becomes a dynamic probabilistic network. For inference, messages are iteratively passed locally within each slice. This is followed by message passing across the time slices in the forward direction and then in the backward direction. Accounting

for temporal dependencies thus leads to temporal smoothing of the soft decisions within each shot.

## 6  Results

We compare detection performance of the multijects with and without accounting for the concept dependencies and temporal dependencies. The reference system performs multiject detection by thresholding soft-decisions (i.e., $P(F_i|\mathcal{X})$) at the frame-level. The proposed schemes are then evaluated by thresholding the soft decisions obtained after message passing using the structures in Figures 2 (b), (c) (conceptual dependencies) and Figures 3 (a), (b) (conceptual and temporal dependencies). We use receiver operating characteristics (ROC) curves which show a plot of the probability of detection plotted against the probability of false alarms for different values of a parameter (the threshold in our case).

Figure 4 shows the ROC curves for the overall performance over the test-set across all the five multijects. The three curves in Figure 4 (a) correspond to the performance using isolated frame-level classification, the factor graph in Figure 2 (b) and the factor graph in Figure 2 (c) with ten iterations of loopy propagation. The curves in Figure 4 (b) correspond to isolated detection followed by temporal smoothing, the dynamic multinet in Figure 3 (a) and the one in Figure 3 (b) respectively. From

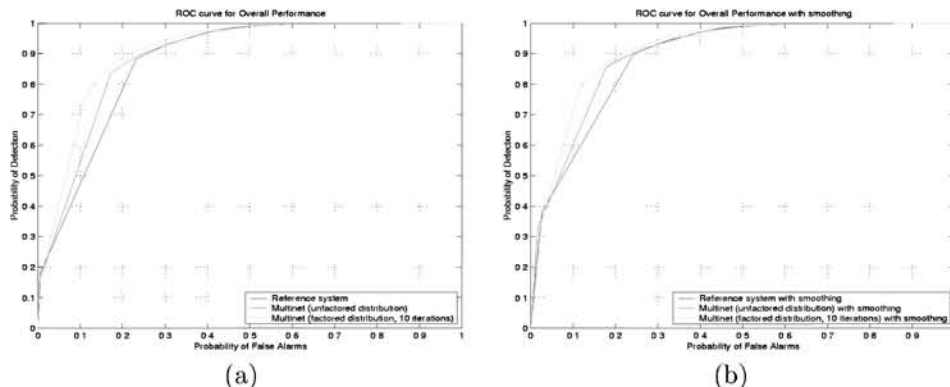

Figure 4: ROC curves for overall performance using isolated detection and two factor graph representations. (a) With static multinets (b) With dynamic multinets.

Figure 4 we observe that there is significant improvement in detection performance by using the multinet to model the dependencies between concepts than without using it. This improvement is especially stark for low $P_f$ where detection rate improves by more than 22 % for a threshold corresponding to $P_f = 0.1$. Interestingly, detection based on the factorized functions (Figure 2 (c)) performs better than the the one based on the unfactorized function. This suggests that the factorized function is a better representative and can be estimated more reliably due to fewer parameters being involved. Also by using models in Figure 3, which account for temporal dependencies across video frames and by performing smoothing using the forward backward algorithm, we see further improvement in detection performance in Figure 4 (b). The detection rate corresponding to $P_f = 0.1$ is 68 % for the static multinet (Figure 2 (c)) and 72 % for its dynamic counterpart (Figure 3 (b)).

Comparison of ROC curves with and without temporal smoothing (not shown here due to lack of space) reveal that temporal smoothing results in better detection irrespective of the threshold or configuration.

## 7 Conclusions and Future Research

We propose a probabilistic framework for detecting semantic concepts using multijects and multinets. We present implementations of static and dynamic multinets using factor graphs. We show that there is significant improvement in detection performance by accounting for the interaction between semantic concepts and temporal dependency amongst the concepts. The multinet architecture imposes no restrictions on the classifiers used in the multijects and we can improve performance by using better multiject models. Our framework can be easily expanded to integrate multiple modalities if they have not been integrated in the multijects to account for the loose coupling between audio and visual streams in movies. It can also support inference of concepts that are observed not through media features but through their relation to those concepts which are observed in media features.

## Footnotes

[1]Automatic feature selection is not addressed here.

[2]We thank Prof. Chang and D. Zhong for the algorithm [2].

[3]$P_0(\vec{X}_j)$ used 5 gaussian components, while $P_1(\vec{X}_j)$ used 10. The number of mixing components can be fixed experimentally and could be different for optimal performance. In general models for $H_0$ are represented better with more components than those for $H_1$

## References

[1] F. Kschischang, B. Frey, and H.-A. Loeliger, "Factor graphs and the sum-product algorithm," *submitted to* IEEE Trans. Inform. Theory, July 1998.

[2] D. Zhong and S. F. Chang, "Spatio-temporal video search using the object-based video representation," in *Proceedings of the IEEE International Conference on Image Processing*, vol. 2, Santa Barbara, CA, Oct. 1997, pp. 21–24.

[3] M. Naphade, T. Kristjansson, B. Frey, and T. S. Huang, "Probabilistic multimedia objects (multijects): A novel approach to indexing and retrieval in multimedia systems," in *Proceedings of the fifth IEEE International Conference on Image Processing*, vol. 3, Chicago, IL, Oct 1998, pp. 536–540.

[4] S. F. Chang, W. Chen, and H. Sundaram, "Semantic visual templates - linking features to semantics," in *Proceedings of the fifth IEEE International Conference on Image Processing*, vol. 3, Chicago, IL, Oct 1998, pp. 531–535.

[5] M. Naphade, R. Mehrotra, A. M. Ferman, J. Warnick, T. S. Huang, and A. M. Tekalp, "A high performance shot boundary detection algorithm using multiple cues," in *Proceedings of the fifth IEEE International Conference on Image Processing*, vol. 2, Chicago, IL, Oct 1998, pp. 884–887.

[6] R. Jain, R. Kasturi, and B. Schunck, *Machine Vision*. MIT Press and McGraw-Hill, 1995.

[7] A. K. Jain and A. Vailaya, "Shape-based retrieval: A case study with trademark image databases," *Pattern Recognition*, vol. 31, no. 9, pp. 1369–1390, 1998.

[8] S. Dudani, K. Breeding, and R. McGhee, "Aircraft identification by moment invariants," *IEEE Trans. on Computers*, vol. C-26, pp. 39–45, Jan 1977.

[9] M. R. Naphade and T. S. Huang, "A probabilistic framework for semantic indexing and retrieval in video," to appear in *IEEE International Conference on Multimedia and Expo*, New York, NY, July 2000. http://www.ifp.uiuc.edu/~milind/cpapers.html

[10] M. R. Naphade and T. S. Huang, "Stochastic modeling of soundtrack for efficient segmentation and indexing of video," in *SPIE IS & T Storage and Retrieval for Multimedia Databases*, vol. 3972, Jan 2000, pp. 168–176.
